# Efficient Estimation of OOMs

**Herbert Jaeger, Mingjie Zhao, Andreas Kolling**
International University Bremen
Bremen, Germany
`h.jaeger|m.zhao|a.kolling@iu-bremen.de`

## Abstract

A standard method to obtain stochastic models for symbolic time series is to train state-emitting hidden Markov models (SE-HMMs) with the Baum-Welch algorithm. Based on observable operator models (OOMs), in the last few months a number of novel learning algorithms for similar purposes have been developed: (1,2) two versions of an "efficiency sharpening" (ES) algorithm, which iteratively improves the statistical efficiency of a sequence of OOM estimators, (3) a constrained gradient descent ML estimator for transition-emitting HMMs (TE-HMMs). We give an overview on these algorithms and compare them with SE-HMM/EM learning on synthetic and real-life data.

## 1   Introduction

Stochastic symbol sequences with memory effects are frequently modelled by training hidden Markov models with the Baum-Welch variant of the EM algorithm. More specifically, state-emitting HMMs (SE-HMMs) are standardly employed, which emit observable events from hidden states. Known weaknesses of HMM training with Baum-Welch are long runtimes and proneness to getting trapped in local maxima.

Over the last few years, an alternative to HMMs has been developed, *observable operator models* (OOMs). The class of processes that can be described by (finite-dimensional) OOMs properly includes the processes that can be described by (finite-dimensional) HMMs. OOMs identify the observable events $a$ of a process with linear *observable operators* $\tau_a$ acting on a real vector space of *predictive states* $w$ [1]. A basic learning algorithm for OOMs [2] estimates the observable operators $\tau_a$ by solving a linear system of *learning equations*. The learning algorithm is constructive, fast and yields asymptotically correct estimates. Two problems that so far prevented OOMs from practical use were (i) poor statistical efficiency, (ii) the possibility that the obtained models might predict negative "probabilities" for some sequences. Since a few months the first problem has been very satisfactorily solved [2]. In this novel approach to learning OOMs from data we iteratively construct a sequence of estimators whose statistical efficiency increases, which led us to call the method *efficiency sharpening* (ES).

Another, somewhat neglected class of stochastic models is transition-emitting HMMs (TE-HMMs). TE-HMMs fall in between SE-HMMs and OOMs w.r.t. expressiveness. TE-HMMs are equivalent to OOMs whose operator matrices are non-negative. Because TE-HMMs are frequently referred to as Mealy machines (actually a misnomer because orig-

inally Mealy machines are not probabilistic but only non-deterministic), we have started to call non-negative OOMs "Mealy OOMs" (MOOMs). We use either name according to the way the models are represented. A variant of Baum-Welch has recently been described for TE-HMMs [3]. We have derived an alternative learning constrained log gradient (CLG) algorithm for MOOMs which performs a constrained gradient descent on the log likelihood surface in the log model parameter space of MOOMs.

In this article we give a compact introduction to the basics of OOMs (Section 2), outline the new ES and CLG algorithms (Sections 3 and 4), and compare their performance on a variety of datasets (Section 5). In the conclusion (Section 6) we also provide a pointer to a Matlab toolbox.

## 2  Basics of OOMs

Let $(\Omega, \mathfrak{A}, P, (X_n)_{n \geq 0})$ or $(X_n)$ for short be a discrete-time stochastic process with values in a finite symbol set $O = \{a_1, \ldots, a_M\}$. We will consider only stationary processes here for notational simplicity; OOMs can equally model nonstationary processes. An $m$-dimensional OOM for $(X_n)$ is a structure $\mathcal{A} = (\mathbb{R}^m, (\tau_a)_{a \in O}, w_0)$, where each *observable operator* $\tau_a$ is a real-valued $m \times m$ matrix and $w_0 \in \mathbb{R}^m$ is the *starting state*, provided that for any finite sequence $a_{i_0} \ldots a_{i_n}$ it holds that

$$P(X_0 = a_{i_0}, \ldots X_n = a_{i_n}) = \mathbf{1}_m \tau_{a_{i_n}} \cdots \tau_{a_{i_0}} w_0, \tag{1}$$

where $\mathbf{1}_m$ always denotes a row vector of units of length $m$ (we drop the subscript if it is clear from the context). We will use the shorthand notation $\bar{a}$ to denote a generic sequence and $\tau_{\bar{a}}$ to denote a concatenation of the corresponding operators in reverse order, which would condense (1) into $P(\bar{a}) = \mathbf{1} \tau_{\bar{a}} w_0$.

Conversely, if a structure $\mathcal{A} = (\mathbb{R}^m, (\tau_a)_{a \in O}, w_0)$ satisfies

$$\text{(i)} \ \ \mathbf{1} w_0 = 1, \quad \text{(ii)} \ \ \mathbf{1}\Big(\sum_{a \in O} \tau_a\Big) = \mathbf{1}, \quad \text{(iii)} \ \ \forall \bar{a} \in O^* : \ \mathbf{1} \tau_{\bar{a}} w_0 \geq 0, \tag{2}$$

(where $O^*$ denotes the set of all finite sequences over $O$), then there exists a process whose distribution is described by $\mathcal{A}$ via (1). The process is stationary iff $(\sum_{a \in O} \tau_a) w_0 = w_0$. Conditions (i) and (ii) are easy to check, but no efficient criterium is known to decide whether the non-negativity criterium (iii) holds for a structure $\mathcal{A}$ (for recent progress in this problem, which is equivalent to a problem of general interest in linear algebra, see [4]). Models $\mathcal{A}$ learnt from data tend to marginally violate (iii) – this is the unresolved non-negativity problem in the theory of OOMs.

The *state* $w_{\bar{a}}$ of an OOM after an initial history $\bar{a}$ is obtained by normalizing $\tau_{\bar{a}} w_0$ to unit component sum via $w_{\bar{a}} = \tau_{\bar{a}} w_0 / \mathbf{1} \tau_{\bar{a}} w_0$.

A fundamental (and nontrivial) theorem for OOMs characterizes equivalence of two OOMs. Two $m$-dimensional OOms $\mathcal{A} = (\mathbb{R}^m, (\tau_a)_{a \in O}, w_0)$ and $\tilde{\mathcal{A}} = (\mathbb{R}^m, (\tilde{\tau}_a)_{a \in O}, \tilde{w}_0)$ are defined to be equivalent if they generate the same probability distribution according to (1). By the equivalence theorem, $\mathcal{A}$ is equivalent to $\tilde{\mathcal{A}}$ if and only if there exists a transformation matrix $\varrho$ of size $m \times m$, satisfying $\mathbf{1}\varrho = \mathbf{1}$, such that $\tilde{\tau}_a = \varrho \tau_a \varrho^{-1}$ for all symbols $a$.

We mentioned in the Introduction that OOM states represent the future probability distribution of the process. This can be algebraically captured in the notion of *characterizers*. Let $\mathcal{A} = (\mathbb{R}^m, (\tau_a)_{a \in O}, w_0)$ be an OOM for $(X_n)$ and choose $k$ such that $\kappa = |O|^k \geq m$. Let

$\bar{b}_1, \ldots, \bar{b}_\kappa$ be the alphabetical enumeration of $O^k$. Then a $m \times \kappa$ matrix $C$ is a *characterizer of length* $k$ for $\mathcal{A}$ iff $\mathbf{1}C = \mathbf{1}$ (that is, $C$ has unit column sums) and

$$\forall \bar{a} \in O^* : \; w_{\bar{a}} = C(P(\bar{b}_1|\bar{a}) \cdots P(\bar{b}_\kappa|\bar{a}))', \tag{3}$$

where $'$ denotes the transpose and $P(\bar{b}|\bar{a})$ is the conditional probability that the process continues with $\bar{b}$ after an initial history $\bar{a}$. It can be shown [2] that every OOM has characterizers of length $k$ for suitably large $k$. Intuitively, a characterizer "bundles" the length $k$ future distribution into the state vector by projection.

If two equivalent OOMs $\mathcal{A}, \tilde{\mathcal{A}}$ are related by $\tilde{\tau}_a = \varrho \tau_a \varrho^{-1}$, and $C$ is a characterizer for $\mathcal{A}$, it is easy to check that $\varrho C$ is a characterizer for $\tilde{\mathcal{A}}$.

We conclude this section by explaining the basic learning equations. An analysis of (1) reveals that for any state $w_{\bar{a}}$ and operator $\tau_b$ from an OOM it holds that

$$\tau_a w_{\bar{a}} = P(a|\bar{a}) w_{\bar{a}a}, \tag{4}$$

where $\bar{a}a$ is the concatenation of $\bar{a}$ with $a$. The vectors $w_{\bar{a}}$ and $P(a|\bar{a})w_{\bar{a}a}$ thus form an argument-value pair for $\tau_a$. Let $\bar{a}_1, \ldots, \bar{a}_l$ be a finite sequence of finite sequences over $O$, and let $V = (w_{\bar{a}_1} \cdots w_{\bar{a}_l})$ be the matrix containing the corresponding state vectors. Let again $C$ be a $m \times \kappa$ sized characterizer of length $k$ and $\bar{b}_1, \ldots, \bar{b}_\kappa$ be the alphabetical enumeration of $O^k$. Let $\underline{V} = (P(\bar{b}_i|\bar{a}_j))$ be the $\kappa \times l$ matrix containing the conditional continuation probabilities of the initial sequences $\bar{a}_j$ by the sequences $\bar{b}_i$. It is easy to see that $V = C\underline{V}$. Likewise, let $W_a = (P(a|\bar{a}_1)w_{\bar{a}_1a} \cdots P(a|\bar{a}_l)w_{\bar{a}_la})$ contain the vectors corresponding to the rhs of (4), and let $\underline{W}_a = (P(a\bar{b}_i|\bar{a}_j))$ be the analog of $\underline{V}$. It is easily verified that $W_a = C\underline{W}_a$. Furthermore, by construction it holds that $\tau_a V = W_a$.

A linear operator on $\mathbb{R}^m$ is uniquely determined by $l \geq m$ argument-value pairs provided there are at least $m$ linearly independent argument vectors in these pairs. Thus, if a characterizer $C$ is found such that $V = C\underline{V}$ has rank $m$, the operators $\tau_a$ of an OOM characterized by $C$ are uniquely determined by $\underline{V}$ and the matrices $\underline{W}_a$ via $\tau_a = W_a V^\dagger = C\underline{W}_a(C\underline{V})^\dagger$, where $\dagger$ denotes the pseudo-inverse. Now, given a training sequence $S$, the conditional continuation probabilities $P(\bar{b}_i|\bar{a}_j), P(a\bar{b}_i|\bar{a}_j)$ that make up $\underline{V}, \underline{W}_a$ can be estimated from $S$ by an obvious counting scheme, yielding estimates $\hat{P}(\bar{b}_i|\bar{a}_j), \hat{P}(a\bar{b}_i|\bar{a}_j)$ for making up $\hat{\underline{V}}$ and $\hat{\underline{W}}_a$, respectively. This leads to the general form of OOM learning equations:

$$\hat{\tau}_a = C\hat{\underline{W}}_a(C\hat{\underline{V}})^\dagger. \tag{5}$$

In words, to learn an OOM from $S$, first fix a model dimension $m$, a characterizer $C$, *indicative sequences* $\bar{a}_1, \ldots, \bar{a}_l$, then construct estimates $\hat{\underline{V}}$ and $\hat{\underline{W}}_a$ by frequency counting, and finally use (5) to obtain estimates of the operators. This estimation procedure is asymptotically correct in the sense that, if the training data were generated by an $m$-dimensional OOM in the first place, this generator will almost surely be perfectly recovered as the size of training data goes to infinity. The reason for this is that the estimates $\hat{\underline{V}}$ and $\hat{\underline{W}}_a$ converge almost surely to $\underline{V}$ and $\underline{W}_a$. The starting state can be recovered from the estimated operators by exploiting $(\sum_{a \in O} \tau_a)w_0 = w_0$ or directly from $C$ and $\hat{\underline{V}}$ (see [2] for details).

## 3 The ES Family of Learning Algorithms

All learning algorithms based on (5) are asymptotically correct (which EM algorithms are not, by the way), but their statistical efficiency (model variance) depends crucially on (i)

the choice of indicative sequences $\bar{a}_1, \ldots, \bar{a}_l$ and (ii) the characterizer $C$ (assuming that the model dimension $m$ is determined by other means, e.g. by cross-validation). We will first address (ii) and describe an iterative scheme to obtain characterizers that lead to a low model variance.

The choice of $C$ has a twofold impact on model variance. First, the pseudoinverse operation in (5) blows up variation in $C\hat{\underline{V}}$ depending on the matrix condition number of this matrix. Thus, $C$ should be chosen such that the condition of $C\hat{\underline{V}}$ gets close to 1. This strategy was pioneered in [5], who obtained the first halfway statistically satisfactory learning procedures. In contrast, here we set out from the second mechanism by which $C$ influences model variance, namely, choose $C$ such that the variance of $C\hat{\underline{V}}$ itself is minimized.

We need a few algebraic preparations. First, observe that if some characterizer $C$ is used with (5), obtaining a model $\hat{\mathcal{A}}$, and $\varrho$ is an OOM equivalence transformation, then if $\tilde{C} = \varrho C$ is used with (5), the obtained model $\tilde{\hat{\mathcal{A}}}$ is an equivalent version of $\hat{\mathcal{A}}$ via $\varrho$.

Furthermore, it is easy to see [2] that two characterizers $C_1, C_2$ characterize the same OOM iff $C_1\underline{V} = C_2\underline{V}$. We call two characterizers *similar* if this holds, and write $C_1 \sim C_2$. Clearly $C_1 \sim C_2$ iff $C_2 = C_1 + G$ for some $G$ satisfying $G\underline{V} = \mathbf{0}$ and $\mathbf{1}G = \mathbf{0}$. That is, the similarity equivalence class of some characterizer $C$ is the set $\{C + G | G\underline{V} = \mathbf{0}, \mathbf{1}G = \mathbf{0}\}$. Together with the first observation this implies that we may confine our search for "good" characterizers to a single (and arbitrary) such equivalence class of characterizers. Let $C_0$ in the remainder be a representative of an arbitrarily chosen similarity class whose members all characterize $\mathcal{A}$.

In [2] it is explained that the variance of $C\hat{\underline{V}}$ is monotonically tied to $\sum_{i=1,\ldots,\kappa; j=1,\ldots,l} P(\bar{a}_j\bar{b}_i)\|w_{\bar{a}_j} - C(:,i)\|^2$, where $C(:,i)$ is the $i$-th column of $C$. This observation allows us to determine an optimal (minimal variance of $C\hat{\underline{V}}$ within the equivalence class of $C_0$) characterizer $C_{\text{opt}}$ as the solution to the following minimization problem:

$$
\begin{aligned}
C_{\text{opt}} &= C_0 + G_{\text{opt}}, \quad \text{where} \\
G_{\text{opt}} &= \arg\min_G \sum_{i=1,\ldots,\kappa; j=1,\ldots,l} P(\bar{a}_j\bar{b}_i)\|w_{\bar{a}_j} - (C_0 + G)(:,i)\|^2
\end{aligned}
\tag{6}
$$

under the constraints $G\underline{V} = \mathbf{0}$ and $\mathbf{1}G = \mathbf{0}$. This problem can be analytically solved [2] and has a surprising and beautiful solution, which we now explain. In a nutshell, $C_{\text{opt}}$ is composed column-wise by certain states of a time-reversed version of $\mathcal{A}$. We describe in more detail time-reversal of OOMs. Given an OOM $\mathcal{A} = (\mathbb{R}^m, (\tau_a)_{a\in O}, w_0)$ with an induced probability distribution $P_{\mathcal{A}}$, its *reverse* OOM $\mathcal{A}^r = (\mathbb{R}^m, (\tau_a^r)_{a\in O}, w_0^r)$ is characterized by a probability distribution $P_{\mathcal{A}^r}$ satisfying

$$
\forall\, a_0 \cdots a_n \in O^* : \quad P_{\mathcal{A}}(a_0 \cdots a_n) = P_{\mathcal{A}^r}(a_n \cdots a_0).
\tag{7}
$$

A reverse OOM can be easily computed from the "forward" OOM as follows. If $\mathcal{A} = (\mathbb{R}^m, (\tau_a)_{a\in O}, w_0)$ is an OOM for a stationary process, and $w_0$ has no zero entry, then

$$
\mathcal{A}^r = (\mathbb{R}^m, (D\tau_a'D^{-1})_{a\in O}, w_0)
\tag{8}
$$

is a reverse OOM to $\mathcal{A}$, where $D = \text{diag}(w_0)$ is a diagonal matrix with $w_0$ on its diagonal.

Now let $\bar{b}_1, \ldots, \bar{b}_\kappa$ again be the sequences employed in $\underline{V}$. Let $\mathcal{A}^r = (\mathbb{R}^m, (\tau_a^r)_{a\in O}, w_0)$ be the reverse OOM to $\mathcal{A}$, which was characterized by $C_0$. Furthermore, for $\bar{b}_i = b_1 \ldots b_k$ let

$w_{\bar{b}_i}^r = \tau_{b_1}^r \cdots \tau_{b_k}^r w_0 / \mathbf{1}\tau_{b_1}^r \cdots \tau_{b_k}^r w_0$. Then it holds that $C = (w_{\bar{b}_1}^r \cdots w_{\bar{b}_\kappa}^r)$ is a characterizer for an OOM equivalent to $\mathcal{A}$. $C$ can effectively be transformed into a characterizer $C^r$ for $\mathcal{A}$ by $C^r = \varrho^r C$, where

$$\varrho^r = (C \begin{pmatrix} \mathbf{1}\tau_{\bar{b}_1} \\ \vdots \\ \mathbf{1}\tau_{\bar{b}_\kappa} \end{pmatrix})^{-1}. \tag{9}$$

We call $C^r$ the *reverse characterizer* of $\mathcal{A}$, because it is composed from the states of a reverse OOM to $\mathcal{A}$. The analytical solution to (6) turns out to be [2]

$$C_{\text{opt}} = C^r. \tag{10}$$

To summarize, within a similarity class of characterizers, the one which minimizes model variance is the (unique) reverse characterizer in this class. It can be cheaply computed from the "forward" OOM via (8) and (9). This analytical finding suggests the following generic, iterative procedure to obtain characterizers that minimize model variance:

1. **Setup.** Choose a model dimension $m$ and a characterizer length $k$. Compute $\underline{V}, \underline{W}_a$ from the training string $S$.

2. **Initialization.** Estimate an initial model $\hat{\mathcal{A}}^{(0)}$ with some "classical" OOM estimation method (a refined such method is detailed out in [2]).

3. **Efficiency sharpening iteration.** Assume that $\hat{\mathcal{A}}^{(n)}$ is given. Compute its reverse characterizer $\hat{C}^{r(n+1)}$. Use this in (5) to obtain a new model estimate $\hat{\mathcal{A}}^{(n+1)}$.

4. **Termination.** Terminate when the training log-likelihood of models $\hat{\mathcal{A}}^{(n)}$ appear to settle on a plateau.

The rationale behind this scheme is that the initial model $\hat{\mathcal{A}}^{(0)}$ is obtained essentially from an uninformed, ad hoc characterizer, for which one has to expect a large model variation and thus (on the average) a poor $\hat{\mathcal{A}}^{(0)}$. However, the characterizer $\hat{C}^{r(1)}$ obtained from the reversed $\hat{\mathcal{A}}^{(0)}$ is not uninformed any longer but shaped by a reasonable reverse model. Thus the estimator producing $\hat{\mathcal{A}}^{(1)}$ can be expected to produce a model closer to the correct one due to its improved efficiency, etc. Notice that this does not guarantee a convergence of models, nor any monotonic development of any performance parameter in the obtained model sequence. In fact, the training log likelihood of the model sequence typically shoots to a plateau level in about 2 to 5 iterations, after which it starts to jitter about this level, only slowly coming to rest – or even not stabilizing at all; it is sometimes observed that the log likelihood enters a small-amplitude oscillation around the plateau level. An analytical understanding of the asymptotic learning dynamics cannot currently be offered.

We have developed two specific instantiations of the general ES learning scheme, differentiated by the set of indicative sequences used. The first simply uses $l = \kappa, \bar{a}_1, \ldots, \bar{a}_l = \bar{b}_1, \ldots, \bar{b}_\kappa$, which leads to a computationally very cheap iterated recomputation of (5) with updated reverse characterizers. We call this the "poor man's" ES algorithm.

The statistical efficiency of the poor man's ES algorithm is impaired by the fact that only the counting statistics of subsequences of length $2k$ are exploited. The other ES instantiation exploits the statistics of *all* subsequences in the original training string. It is technically rather involved and rests on a suffix tree (ST) representation of $S$. We can only give a coarse sketch here (details in [2]). In each iteration, the current reverse model is run backwards through $S$ and the obtained reverse states are additively collected bottom-up in the

nodes of the ST. From the ST nodes the collected states are then harvested into matrices corresponding directly to $C\hat{\underline{V}}$ and $C\hat{\underline{W}}_a$, that is, an explicit computation of the reverse characterizer is not required. This method incurs a computational load per iteration which is somewhat lower than Baum-Welch for SE-HMMs (because only a backward pass of the current model has to be computed), plus the required initial ST construction which is linear in the size of $S$.

## 4  The CLG Algorithm

We must be very brief here due to space limitations. The CLG algorithm will be detailed out in a future paper. It is an iterative update scheme for the matrix parameters $[\hat{\tau}_a]_{ij}$ of a MOOM. This scheme is analytically derived as gradient descent in the model log likelihood surface over the log space of these matrix parameters, observing constraints of non-negativity of these parameters and the general OOM constraints (i) and (ii) from Eqn. (2). Note that the constraint (iii) from (2) is automatically satisfied in MOOMs.

We skip the derivation of the CLG scheme and describe only its "mechanics". Let $S = s_1 \ldots s_N$ be the training string and for $1 \leq k \leq N$ define $\bar{a}_k = s_1 \ldots s_k, \bar{b}_k = s_{k+1} \ldots s_N$. Define for some $m$-dimensional OOM and $a \in O$

$$\sigma_k = \frac{\mathbf{1}\tau_{\bar{b}_k}}{\mathbf{1}\tau_{\bar{b}_k} w_{\bar{a}_k}}, \quad y_a = \sum_{s_k = a} \frac{\sigma'_k w'_{\bar{a}_{k-1}}}{\mathbf{1}\tau_{s_k} w_{\bar{a}_{k-1}}}, \quad y_0 = \max_{i,j,a}\{[y_a]_{ij}\}, \quad [y_{a0}]_{i,j} = [y_a]_{i,j}/y_0. \tag{11}$$

Then the update equation is

$$[\hat{\tau}_a^+]_{ij} = \eta_j \cdot [\hat{\tau}_a]_{ij} \cdot [y_{a0}]_{ij}^\lambda, \tag{12}$$

where $\hat{\tau}_a^+$ is the new estimate of $\tau_a$, $\eta_j$'s are normalization parameters determined by the constraint (ii) from Eqn. (2), and $\lambda$ is a learning rate which here unconventionally appears in the exponent because the gradient descent is carried out in the log parameter space. Note that by (12) $[\hat{\tau}_a^+]_{ij}$ remains non-negative if $[\hat{\tau}_a]_{ij}$ is. This update scheme is derived in a way that is unrelated to the derivation of the EM algorithm; to our surprise we found that for $\lambda = 1$ (12) is equivalent to the Baum-Welch algorithm for TE-HMMs. However, significantly faster convergence is achieved with non-unit $\lambda$; in the experiments carried out so far a value close to 2 was heuristically found to work best.

## 5  Numerical Comparisons

We compared the poor man's ES algorithm, the suffix-tree based algorithm, the CLG algorithm and the standard SE-HMM/Baum-Welch method on four different types of data, which were generated by (a) randomly constructed, 10-dimensional, 5-symbol SE-HMMs, (b) randomly constructed, 10-dimensional, 5-symbol MOOMs, (c) a 3-dimensional, 2-symbol OOM which is not equivalent to any HMM nor MOOM (the "probability clock" process [2]), (d) a belletristic text (Mark Twain's short story "The 1,000,000 Pound Note"). For each of (a) and (b), 40 experiments were carried out with freshly constructed generators per experiment; a training string of length 1000 and a test string of length 10000 was produced from each generator. For (c), likewise 40 experiments were carried out with freshly generated training/testing sequences of same lengthes as before; here however the generator was identical for all experiments. For (a) – (c), the results reported below are averaged numbers over the 40 experiments. For the (d) dataset, after preprocessing which

shrunk the number of different symbols to 27, the original string was sorted sentence-wise into a training and a testing string, each of length $\sim 21000$ (details in [2]).

The following settings were used with the various training methods. (i) The poor man's ES algorithm was used with a length $k = 2$ of indicative sequences on all datasets. Two ES iterations were carried out and the model of the last iteration was used to compute the reported log likelihoods. (ii) For the suffix-tree based ES algorithm, on datasets (a) – (c), likewise two ES iterations were done and the model from the iteration with the lowest (reverse) training LL was used for reporting. On dataset (d), 4 ES iterations were called and similarly the model with the best reverse training LL was chosen. (iii) In the MOOM studies, a learning rate of $\lambda = 1.85$ was used. Iterations were stopped when two consecutive training LL's differed by less than 5e-5% or after 100 iterations. (iv) For HMM/Baum-Welch training, the public-domain implementation provided by Kevin Murphy was used. Iterations were stopped after 100 steps or if LL's differed by less than 1e-5%. All computations were done in Matlab on 2 GHz PCs except the HMM training on dataset (d) which was done on a 330 MHz machine (the reported CPU times were scaled by 330/2000 to make them comparable with the other studies). Figure 1 shows the training and testing loglikelihoods as well as the CPU times for all methods and datasets.

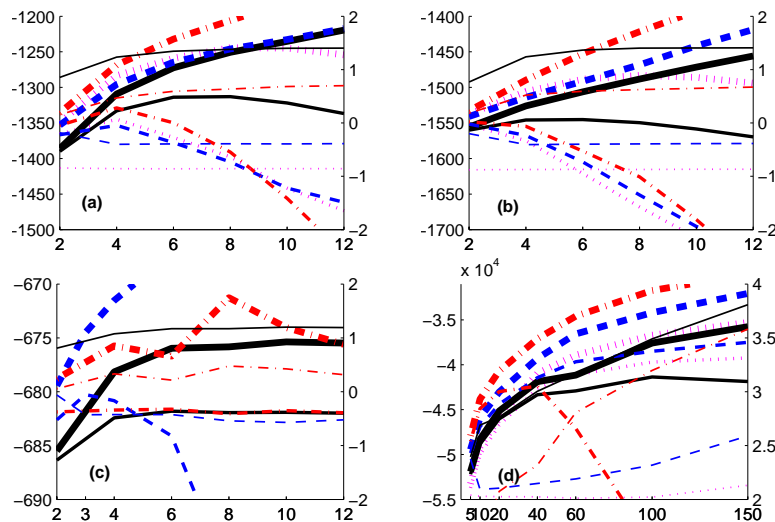

Figure 1: Findings for datasets (a)–(d). In each panel, the left $y$-axis shows log likelihoods for training and testing (testing LL normalized to training stringlength) and the right $y$-axis measures the log 10 of CPU times. HMM models are documented in solid/black lines, poor man's ES models in dotted/magenta lines, suffix-tree ES models in broken/blue, and MOOMs in dash-dotted/red lines. The thickest lines in each panel show training LL, the thinnest CPU time, and intermediate testing LL. The $x$-axes indicate model dimension. On dataset (c), no results of the poor man's algorithm are given because the learning equations became ill-conditioned for all but the lowest dimensions.

Some comments on Fig. 1. (1) The CPU times roughly exhibit an even log spread over almost 2 orders of magnitude, in the order poor man's (fastest) – suffix-tree ES – CLG – Baum-Welch. (2) CLG has the lowest training LL throughout, which needs an explanation because the proper OOMs trained by ES are more expressive. Apparently the ES algorithm does not lead to local ML optima; otherwise suffix-tree ES models should show the lowest training LL. (3) On HMM-generated data (a), Baum-Welch HMMs can play out their

natural bias for this sort of data and achieve a lower test error than the other methods. (4) On the MOOM data (b), the test LL of MOOM/CLG and OOM/poor man models of dimension 2 equals the best HMM/Baum-Welch test LL which is attained at a dimension of 4; the OOM/suffix-tree test LL at dimension 2 is superior to the best HMM test LL. (5) On the "probability clock" data (c), the suffix-tree ES trained OOMs surpassed the non-OOM models in test LL, with the optimal value obtained at the (correct) model dimension 3. This comes as no surprise because these data come from a generator that is incommensurable with either HMMs or MOOMs. (6) On the large empirical dataset (d) the CLG/MOOMs have by a fair margin the highest training LL, but the test LL quickly drops to unacceptable lows. It is hard to explain this by overfitting, considering the complexity and the size of the training string. The other three types of models are evenly ordered in both training and testing error from HMMs (poorest) to suffix-tree ES trained OOMs. Overfitting does not occur up to the maximal dimension investigated. Depending on whether one wants a very fast algorithm with good, or a fast algorithm with very good train/test LL, one here would choose the poor man's or the suffix-tree ES algorithm as the winner. (7) One detail in panel (d) needs an explanation. The CPU time for the suffix-tree ES has an isolated peak for the smallest dimension. This is earned by the construction of the suffix tree, which was built only for the smallest dimension and re-used later.

## 6   Conclusion

We presented, in a sadly condensed fashion, three novel learning algorithms for symbol dynamics. A detailed treatment of the Efficiency Sharpening algorithm is given in [2], and a Matlab toolbox for it can be fetched from http://www.faculty.iu-bremen.de/hjaeger/OOM/OOMTool.zip. The numerical investigations reported here were done using this toolbox. Our numerical simulations demonstrate that there is an altogether new world of faster and often statistically more efficient algorithms for sequence modelling than Baum-Welch/SE-HMMs. The topics that we will address next in our research group are (i) a mathematical analysis of the asymptotic behaviour of the ES algorithms, (ii) online adaptive versions of these algorithms, and (iii) versions of the ES algorithms for nonstationary time series.

## References

[1] M. L. Littman, R. S. Sutton, and S. Singh. Predictive representation of state. In *Advances in Neural Information Processing Systems 14 (Proc. NIPS 01)*, pages 1555–1561, 2001. http://www.eecs.umich.edu/∼baveja/Papers/psr.pdf.

[2] H. Jaeger, M. Zhao, K. Kretzschmar, T. Oberstein, D. Popovici, and A. Kolling. Learning observable operator models via the es algorithm. In S. Haykin, J. Principe, T. Sejnowski, and J. McWhirter, editors, *New Directions in Statistical Signal Processing: from Systems to Brains*, chapter 20. MIT Press, to appear in 2005.

[3] H. Xue and V. Govindaraju. Stochastic models combining discrete symbols and continuous attributes in handwriting recognition. In *Proc. DAS 2002*, 2002.

[4] R. Edwards, J.J. McDonald, and M.J. Tsatsomeros. On matrices with common invariant cones with applications in neural and gene networks. *Linear Algebra and its Applications*, in press, 2004 (online version). http://www.math.wsu.edu/math/faculty/tsat/files/emt.pdf.

[5] K. Kretzschmar. Learning symbol sequences with Observable Operator Models. GMD Report 161, Fraunhofer Institute AIS, 2003. http://omk.sourceforge.net/files/OomLearn.pdf.
